# Categorization by Learning and Combining Object Parts

**Bernd Heisele**[†‡] **Thomas Serre**[†] **Massimiliano Pontil**[§] **Thomas Vetter**[*] **Tomaso Poggio**[†]
[†]Center for Biological and Computational Learning, M.I.T., Cambridge, MA, USA
[‡]Honda R&D Americas, Inc., Boston, MA, USA
[§]Department of Information Engineering, University of Siena, Siena, Italy
[*]Computer Graphics Research Group, University of Freiburg, Freiburg, Germany
{*heisele,serre,tp*}@*ai.mit.edu    pontil@ing.unisi.it    vetter@informatik.uni-freiburg.de*

## Abstract

We describe an algorithm for automatically learning discriminative components of objects with SVM classifiers. It is based on growing image parts by minimizing theoretical bounds on the error probability of an SVM. Component-based face classifiers are then combined in a second stage to yield a hierarchical SVM classifier. Experimental results in face classification show considerable robustness against rotations in depth and suggest performance at significantly better level than other face detection systems. Novel aspects of our approach are: a) an algorithm to learn component-based classification experts and their combination, b) the use of 3-D morphable models for training, and c) a maximum operation on the output of each component classifier which may be relevant for biological models of visual recognition.

## 1   Introduction

We study the problem of automatically synthesizing hierarchical classifiers by learning discriminative object parts in images. Our motivation is that most object classes (e.g. faces, cars) seem to be naturally described by a few characteristic parts or components and their geometrical relation. Greater invariance to viewpoint changes and robustness against partial occlusions are the two main potential advantages of component-based approaches compared to a global approach.

The first challenge in developing component-based systems is how to choose automatically a set of discriminative object components. Instead of manually selecting the components, it is desirable to learn the components from a set of examples based on their discriminative power and their robustness against pose and illumination changes. The second challenge is to combine the component-based experts to perform the final classification.

## 2  Background

Global approaches in which the whole pattern of an object is used as input to a single classifier were successfully applied to tasks where the pose of the object was fixed. In [6] Haar wavelet features are used to detect frontal and back views of pedestrians with an SVM classifier. Learning-based systems for detecting frontal faces based on a gray value features are described in [14, 13, 10, 2].

Component-based techniques promise to provide more invariance since the individual components vary less under pose changes than the whole object. Variations induced by pose changes occur mainly in the locations of the components. A component-based method for detecting faces based on the empirical probabilities of overlapping rectangular image parts is proposed in [11]. Another probabilistic approach which detects small parts of faces is proposed in [4]. It uses local feature extractors to detect the eyes, the corner of the mouth, and the tip of the nose. The geometrical configuration of these features is matched with a model configuration by conditional search. A related method using statistical models is published in [9]. Local features are extracted by applying multi-scale and multi-orientation filters to the input image. The responses of the filters on the training set are modeled as Gaussian distributions. In [5] pedestrian detection is performed by a set of SVM classifiers each of which was trained to detect a specific part of the human body.

In this paper we present a technique for learning relevant object components. The technique starts with a set of small seed regions which are gradually grown by minimizing a bound on the expected error probability of an SVM. Once the components have been determined, we train a system consisting of a two-level hierarchy of SVM classifiers. First, component classifiers independently detect facial components. Second, a combination classifier learns the geometrical relation between the components and performs the final detection of the object.

## 3  Learning Components with Support Vector Machines

### 3.1  Linear Support Vector Machines

Linear SVMs [15] perform pattern recognition for two-class problems by determining the separating hyperplane with maximum distance to the closest points in the training set. These points are called support vectors. The decision function of the SVM has the form:

$$f(\mathbf{x}) = \sum_{i=1}^{\ell} \alpha_i y_i < \mathbf{x}_i \cdot \mathbf{x} > + b,$$

(1)

where $\ell$ is the number of data points and $y_i \in \{-1, 1\}$ is the class label of the data point $x_i$. The coefficients $\alpha_i$ are the solution of a quadratic programming problem. The margin $M$ is the distance of the support vectors to the hyperplane, it is given by:

$$M = \frac{1}{\sqrt{\sum_{i}^{\ell} \alpha_i}}.$$

(2)

The margin is an indicator of the separability of the data. In fact, the expected error probability of the SVM, $EP_{err}$, satisfies the following bound [15]:

$$EP_{err} \leq \frac{1}{\ell} E \left[ \frac{D^2}{M^2} \right],$$

(3)

where $D$ is the diameter of the smallest sphere containing all data points in the training set.

## 3.2  Learning Components

Our method automatically determines rectangular components from a set of object images. The algorithm starts with a small rectangular component located around a pre-selected point in the object image (e.g. for faces this could be the center of the left eye). The component is extracted from each object image to build a training set of positive examples. We also generate a training set of background patterns that have the same rectangular shape as the component. After training an SVM on the component data we estimate the performance of the SVM based on the upper bound on the error probability. According to Eq. (3) we calculate:

$$\rho = \frac{D^2}{M^2}.$$  (4)

As shown in [15] this quantity can be computed by solving a quadratic programming problem. After determining $\rho$ we enlarge the component by expanding the rectangle by one pixel into one of the four directions (up, down, left, right). Again, we generate training data, train an SVM and determine $\rho$. We do this for expansions into all four directions and finally keep the expansion which decreases $\rho$ the most. This process is continued until the expansions into all four directions lead to an increase of $\rho$. In order to learn a set of components this process can be applied to different seed regions.

## 4  Learning Facial Components

Extracting face patterns is usually a tedious and time-consuming work that has to be done manually. Taking the component-based approach we would have to manually extract each single component from all images in the training set. This procedure would only be feasible for a small number of components. For this reason we used textured 3-D head models [16] to generate the training data. By rendering the 3-D head models we could automatically generate large numbers of faces in arbitrary poses and with arbitrary illumination. In addition to the 3-D information we also knew the 3-D correspondences for a set of reference points shown in Fig. 1a). These correspondences allowed us to automatically extract facial components located around the reference points. Originally we had seven textured head models acquired by a 3-D scanner. Additional head models were generated by 3-D morphing between all pairs of the original head models. The heads were rotated between $-30°$ and $30°$ in depth. The faces were illuminated by ambient light and a single directional light pointing towards the center of the face. The position of the light varied between $-30°$ and $30°$ in azimuth and between $30°$ and $60°$ in elevation. Overall, we generated 2,457 face images of size 58×58. Some examples of synthetic face images used for training are shown in Fig. 1b).

The negative training set initially consisted of 10,209 58×58 non-face patterns randomly extracted from 502 non-face images. We then applied bootstrapping to enlarge the training data by non-face patterns that look similar to faces. To do so we trained a single linear SVM classifier and applied it to the previously used set of 502 non-face images. The false positives (FPs) were added to the non-face training data to build the final training set of size 13,654.

We started with fourteen manually selected seed regions of size 5×5. The resulting components were located around the eyes (17×17 pixels), the nose (15×20 pixels), the mouth (31×15 pixels), the cheeks (21×20 pixels), the lip (13×16 pixels), the nostrils (22 × 12 pixels), the corners of the mouth (18×11 pixels), the eyebrows (19×15 pixels), and the bridge of the nose (18×16 pixels).

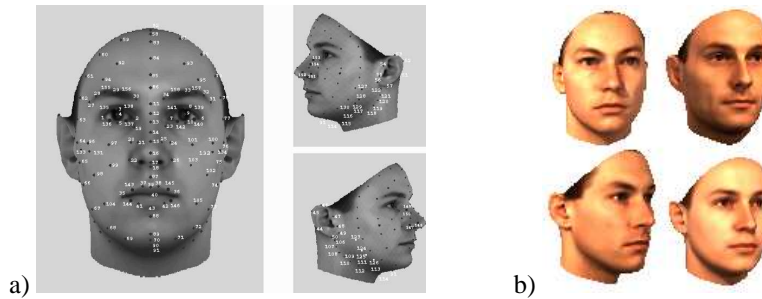

a)                                                                b)

Figure 1: a) Reference points on the head models which were used for 3-D morphing and automatic extraction of facial components. b) Examples of synthetic faces.

## 5  Combining Components

An overview of our two-level component-based classifier is shown in Fig. 2. On the first level the component classifiers independently detect components of the face. Each classifier was trained on a set of facial components and on a set of non-face patterns generated from the training set described in Section 4. On the second level the combination classifier performs the detection of the face based on the outputs of the component classifiers. The maximum real-valued outputs of each component classifier within rectangular search regions around the expected positions of the components are used as inputs to the combination classifier. The size of the search regions was estimated from the mean and the standard deviation of the locations of the components in the training images. The maximum operation is performed both during training and at run-time. Interestingly it turns out to be similar to the key pooling mechanism postulated in a recent model of object recognition in the visual cortex [8]. We also provide the combination classifier with the precise positions of the detected components relative to the upper left corner of the $58{\times}58$ window. Overall we have three values per component classifier that are propagated to the combination classifier: the maximum output of the component classifier and the $x$-$y$ image coordinates of the maximum.

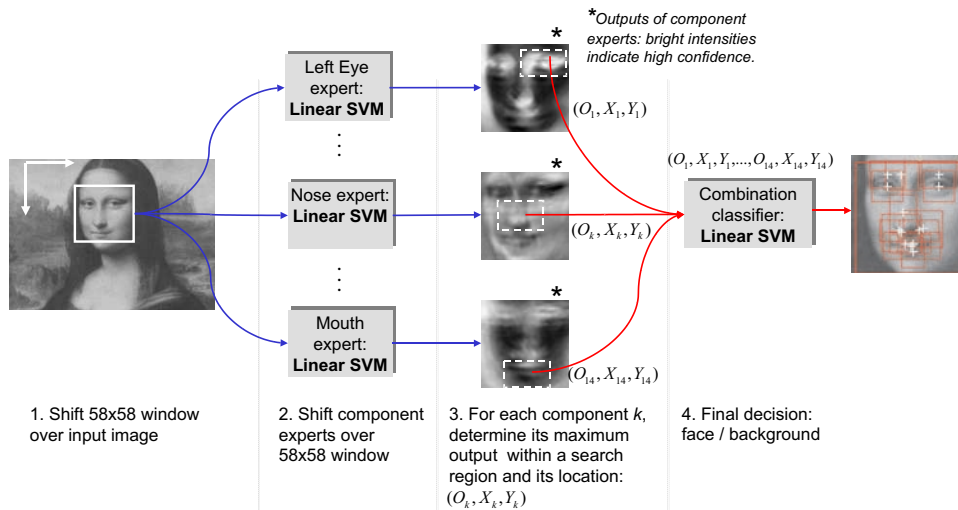

Figure 2: System overview of the component-based classifier.

# 6 Experiments

In our experiments we compared the component-based system to global classifiers. The component system consisted of fourteen linear SVM classifiers for detecting the components and a single linear SVM as combination classifier. The global classifiers were a single linear SVM and a single second-degree polynomial SVM both trained on the gray values of the whole face pattern. The training data for these three classifiers consisted of 2,457 synthetic gray face images and 13,654 non-face gray images of size $58 \times 58$. The positive test set consisted of 1,834 faces rotated between about $-30°$ and $30°$ in depth. The faces were manually extracted from the CMU PIE database [12]. The negative test set consisted of 24,464 difficult non-face patterns that were collected by a fast face detector [3] from web images. The FP rate was calculated relative to the number of non-face test images. Because of the resolution required by the component-based system, a direct comparison with other published systems on the standard MIT-CMU test set [10] was impossible. For an indirect comparison, we used a second-degree polynomial SVM [2] which was trained on a large set of $19 \times 19$ real face images. This classifier performed amongst the best face detection systems on the MIT-CMU test set. The ROC curves in Fig. 3 show that the component-based classifier is significantly better than the three global classifiers. Some detection results generated by the component system are shown in Fig. 4.

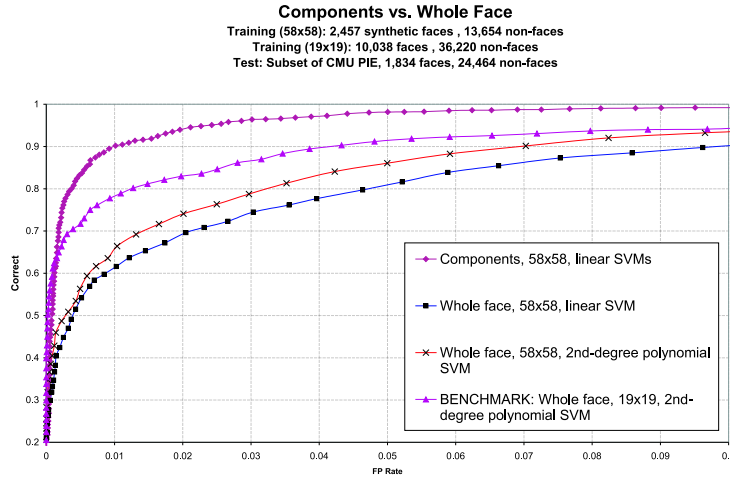

Figure 3: Comparison between global classifiers and the component-based classifier.

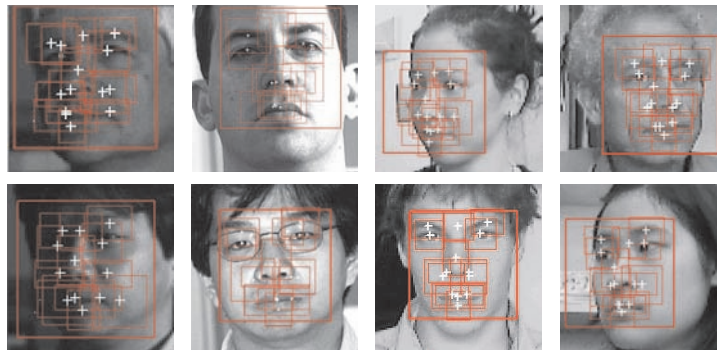

Figure 4: Faces detected by the component-based classifier.

A natural question that arises is about the role of geometrical information. To answer this question–which has relevant implications for models of cortex–we tested another system in which the combination classifier receives as inputs only the output of each component classifier but not the position of its maximum. As shown in Fig. 5 this system still outperforms the whole face systems but it is worse than the system with position information.

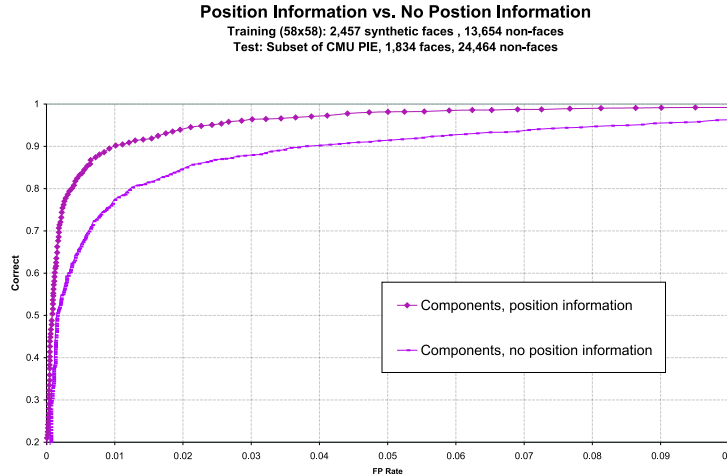

Figure 5: Comparison between a component-based classifier trained with position information and a component-based classifier without position information.

## 7   Open Questions

An extension under way of the component-based approach to face identification is already showing good performances [1]. Another natural generalization of the work described here involves the application of our system to various classes of objects such as cars, animals, and people. Still another extension regards the question of view-invariant object detection. As suggested by [7] in a biological context and demonstrated recently by [11] in machine vision, full pose invariance in recognition tasks can be achieved by combining view-dependent classifiers. It is interesting to ask whether the approach described here could also be used to learn which views are most discriminative and how to combine them optimally. Finally, the role of geometry and in particular how to compute and represent position information in biologically plausible networks, is an important open question at the interface between machine and biological vision.

## References

[1] B. Heisele, P. Ho, and T. Poggio. Face recognition with support vector machines: global versus component-based approach. In *Proc. 8th International Conference on Computer Vision*, Vancouver, 2001.

[2] B. Heisele, T. Poggio, and M. Pontil. Face detection in still gray images. A.I. memo 1687, Center for Biological and Computational Learning, MIT, Cambridge, MA, 2000.

[3] B. Heisele, T. Serre, S. Mukherjee, and T. Poggio. Feature reduction and hierarchy of classifiers for fast object detection in video images. In *Proc. IEEE Conference on Computer Vision and Pattern Recognition*, Hawaii, 2001.

[4] T. K. Leung, M. C. Burl, and P. Perona. Finding faces in cluttered scenes using random labeled graph matching. In *Proc. International Conference on Computer Vision*, pages 637–644, Cambridge, MA, 1995.

[5] A. Mohan, C. Papageorgiou, and T. Poggio. Example-based object detection in images by components. In *IEEE Transactions on Pattern Analysis and Machine Intelligence*, volume 23, pages 349–361, April 2001.

[6] C. Papageorgiou and T. Poggio. A trainable system for object detection. In *International Journal of Computer Vision*, volume 38, 1, pages 15–33, 2000.

[7] T. Poggio and S. Edelman. A network that learns to recognize 3-D objects. *Nature*, 343:163–266, 1990.

[8] M. Riesenhuber and T. Poggio. Hierarchical models of object recognition in cortex. *Nature Neuroscience*, 2(11):1019–1025, 1999.

[9] T. D. Rikert, M. J. Jones, and P. Viola. A cluster-based statistical model for object detection. In *Proc. IEEE Conference on Computer Vision and Pattern Recognition*, volume 2, pages 1046–1053, Fort Collins, 1999.

[10] H. A. Rowley, S. Baluja, and T. Kanade. Rotation invariant neural network-based face detection. Computer Science Technical Report CMU-CS-97-201, CMU, Pittsburgh, 1997.

[11] H. Schneiderman and T. Kanade. A statistical method for 3d object detection applied to faces and cars. In *Proc. IEEE Conference on Computer Vision and Pattern Recognition*, pages 746–751, 2000.

[12] T. Sim, S. Baker, and M. Bsat. The CMU pose, illumination, and expression (PIE) database of human faces. Computer Science Technical Report 01-02, CMU, 2001.

[13] K.-K. Sung. *Learning and Example Selection for Object and Pattern Recognition*. PhD thesis, MIT, Artificial Intelligence Laboratory and Center for Biological and Computational Learning, Cambridge, MA, 1996.

[14] R. Vaillant, C. Monrocq, and Y. Le Cun. An original approach for the localisation of objects in images. In *International Conference on Artificial Neural Networks*, pages 26–30, 1993.

[15] V. Vapnik. *Statistical learning theory*. John Wiley and Sons, New York, 1998.

[16] T. Vetter. Synthesis of novel views from a single face. *International Journal of Computer Vision*, 28(2):103–116, 1998.
